# Analysis of a greedy active learning strategy

**Sanjoy Dasgupta**[*]
University of California, San Diego
dasgupta@cs.ucsd.edu

## Abstract

We abstract out the core search problem of active learning schemes, to better understand the extent to which adaptive labeling can improve sample complexity. We give various upper and lower bounds on the number of labels which need to be queried, and we prove that a popular greedy active learning rule is approximately as good as any other strategy for minimizing this number of labels.

## 1  Introduction

An increasingly common phenomenon in classification tasks is that unlabeled data is abundant, whereas labels are considerably harder to come by. Genome sequencing projects, for instance, are producing vast numbers of peptide sequences, but reliably labeling even one of these with structural information requires time and close attention.

This distinction between labeled and unlabeled data is not captured in standard models like the PAC framework, and has motivated the field of *active learning*, in which the learner is able to ask for the labels of specific points, but is charged for each label. These query points are typically chosen from an unlabeled data set, a practice called *pool-based learning* [10]. There has also been some work on creating query points synthetically, including a rich body of theoretical results [1, 2], but this approach suffers from two problems: first, from a practical viewpoint, the queries thus produced can be quite unnatural and therefore bewildering for a human to classify [3]; second, since these queries are not picked from the underlying data distribution, they might have limited value in terms of generalization. In this paper, we focus on pool-based learning.

We are interested in active learning with generalization guarantees. Suppose the hypothesis class has VC dimension $d$ and we want a classifier whose error rate on distribution $P$ over the joint (input, label) space, is less than $\epsilon > 0$. The theory tells us that in a *supervised* setting, we need some $m = m(\epsilon, d)$ *labeled* points drawn from $P$ (for a fixed level of confidence, which we will henceforth ignore). Can we get away with substantially fewer than $m$ labels if we are given unlabeled points from $P$ and are able to adaptively choose which points to label? How much fewer, and what querying strategies should we follow?

Here is a toy example illustrating the potential of active learning. Suppose the data lie on the real line, and the classifiers are simple thresholding functions, $H = \{h_w : w \in \mathbf{R}\}$:

$$h_w(x) = \mathbf{1}(x \geq w).$$

VC theory tells us that if the underlying distribution $P$ can be classified perfectly by some hypothesis in $H$ (called the *realizable* case), then it is enough to draw $m = O(1/\epsilon)$ random

labeled examples from $P$, and to return any classifier consistent with them. But suppose we instead draw $m$ *unlabeled* samples from $P$. If we lay these points down on the line, their hidden labels are a sequence of 0's followed by a sequence of 1's, and the goal is to discover the point $w$ at which the transition occurs. This can be accomplished with a simple binary search which asks for just $\log m$ labels. Thus active learning gives us an *exponential* improvement in the number of labels needed: by adaptively querying $\log m$ labels, we can automatically infer the rest of them.

**Generalized binary search?**

So far we have only looked at an extremely simple learning problem. For more complicated hypothesis classes $H$, is a sort of a generalized binary search possible? What would the search space look like? For supervised learning, in the realizable case, the usual bounds specify a sample complexity of (very roughly) $m \approx d/\epsilon$ labeled points if the target error rate is $\epsilon$. So let's pick this many unlabeled points, and then try to find a hypothesis consistent with all the hidden labels by adaptively querying just a few of them. We know via Sauer's lemma that $H$ can classify these $m$ points (considered jointly) in at most $O(m^d)$ different ways – in effect, the size of $H$ is reduced to $O(m^d)$. This finite set is the *effective hypothesis class* $\widehat{H}$. (In the 1-d example, $\widehat{H}$ has size $m + 1$, corresponding to the intervals into which the points $x_i$ split the real line.) The most we can possibly learn about the target hypothesis, even if all labels are revealed, is to narrow it down to one of these regions. Is it possible to pick among these $O(m^d)$ possibilities using $o(m)$ labels? If binary search were possible, just $O(d \log m)$ labels would be needed.

Unfortunately, we cannot hope for a generic positive result of this kind. The toy example above is a 1-$d$ linear separator. We show that for $d \geq 2$, the situation is very different:

> Pick any collection of $m$ (unlabeled) points on the unit sphere in $\mathbf{R^d}$, for $d \geq 2$, and assume their hidden labels correspond perfectly to some linear separator. Then there are target hypotheses in $\widehat{H}$ which cannot be identified without querying *all* the labels.

(What if the active learner is not required to identify exactly the right hypothesis, but something close to it? This and other little variations don't help much.) Therefore, even in the most benign situations, we cannot expect that *every* target hypothesis will be identifiable using $o(m)$ labels. To put it differently, in the worst case over target hypotheses, active learning gives no improvement in sample complexity.

But hopefully, *on average* (with respect to some distribution over target hypotheses), the number of labels needed is small. For instance, when $d = 2$ in the bad case above, a target hypothesis chosen uniformly at random from $\widehat{H}$ can be identified by querying just $O(\log m)$ labels in expectation. This motivates the main model of this paper.

**An average-case model**

We will count the expected number of labels queried when the target hypothesis is chosen from some distribution $\pi$ over $\widehat{H}$. This can be interpreted as a Bayesian setting, but it is more accurate to think of $\pi$ merely as a device for averaging query counts, which has no bearing on the final generalization bound. A natural choice is to make $\pi$ uniform over $\widehat{H}$.

Most existing active learning schemes work with $H$ rather than $\widehat{H}$; but $\widehat{H}$ reflects the underlying combinatorial structure of the problem, and it can't hurt to deal with it directly. Often $\pi$ can chosen to mask the structure of $\widehat{H}$; for instance, if $H$ is the set of linear separators, then $\widehat{H}$ is a set of convex regions of $H$, and $\pi$ can be made proportional to the volume of each region. This makes the problem continuous rather than combinatorial.

What is the expected number of labels needed to identify a target hypothesis chosen from $\pi$? In this average-case setting, is it always possible to get away with $o(m)$ labels, where $m$ is the sample complexity of the supervised learning problem as defined above? We show that the answer, once again, is sadly no. Thus the benefit of active learning is really a function of the specific hypothesis class and the particular pool of unlabeled data. Depending on these, the expected number of labels needed lies in the following range (within constants):

| | | |
|---|---|---|
| ideal case: | $d \log m$ | perfect binary search |
| worst case: | $m$ | all labels, or randomly chosen queries |

Notice the exponential gap between the top and bottom of this range. Is there some simple querying strategy which *always* achieves close to the minimum (expected) number of labels, whatever this minimum number might be?

Our main result is that this property holds for a variant of a popular greedy scheme: always ask for the label which most evenly divides the current effective version space weighted by $\pi$. This doesn't necessarily minimize the number of queries, just as a greedy decision tree algorithm need not produce trees of minimum size. However:

> When $\pi$ is uniform over $\widehat{H}$, the expected number of labels needed by this greedy strategy is at most $O(\ln |\widehat{H}|)$ times that of any other strategy.

We also give a bound for arbitrary $\pi$, and show corresponding lower bounds in both the uniform and non-uniform cases.

Variants of this greedy scheme underlie many active learning heuristics, and are often described as optimal in the literature. This is the first rigorous validation of the scheme in a general setting. The performance guarantee is significant: recall $\log |\widehat{H}| = O(d \log m)$, the minimum number of queries possible.

## 2  Preliminaries

Let $\mathcal{X}$ be the input space, $\mathcal{Y} = \{0, 1\}$ the space of labels, and $P$ an unknown underlying distribution over $\mathcal{X} \times \mathcal{Y}$. We want to select a hypothesis (a function $\mathcal{X} \rightarrow \mathcal{Y}$) from some class $H$ of VC dimension $d < \infty$, which will accurately predict labels of points in $\mathcal{X}$. We will assume that the problem is *realizable*, that is, there is some hypothesis in $H$ which gives a correct prediction on every point. Suppose that points $(x_1, y_1) \ldots , (x_m, y_m)$ are drawn randomly from $P$. Standard bounds give us a function $m(\epsilon, d)$ such that if we want a hypothesis of error $\leq \epsilon$ (on $P$, modulo some fixed confidence level), and if $m \geq m(\epsilon, d)$, then we need only pick a hypothesis $h \in H$ consistent with these labeled points [9].

Now suppose just the pool of unlabeled data $x_1, \ldots, x_m$ is available. The possible labelings of these points form a subset of $\{0, 1\}^m$, the *effective hypothesis class*

$$\widehat{H} \cong \{(h(x_1), \ldots, h(x_m)) : h \in H\}.$$

Sauer's lemma [9] tells us $|\widehat{H}| = O(m^d)$. We want to pick the unique $h \in \widehat{H}$ which is consistent with all the hidden labels, by querying just a few of them.

Any deterministic search strategy can be represented as a binary tree whose internal nodes are queries ("what is the $x_i$'s label?"), and whose leaves are elements of $\widehat{H}$. We can also accommodate randomization – for instance, to allow a random choice of query point – by letting internal nodes of the tree be random coin flips. Our main result, Theorem 3, is unaffected by this generalization.

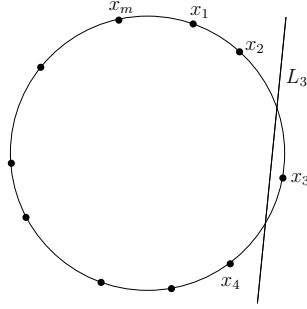

Figure 1: To identify target hypotheses like $L_3$, we need to see *all* the labels.

## 3   Some bad news

**Claim 1** *Let $H$ be the hypothesis class of linear separators in $\mathbf{R}^2$. For any set of $m$ distinct data points on the perimeter of the unit circle, there are always some target hypotheses in $\widehat{H}$ which cannot be identified without querying all $m$ labels.*

*Proof.*  To see this, consider the following realizable labelings (Figure 1):

- Labeling $L_0$: all points are negative.

- Labeling $L_i$ ($1 \leq i \leq m$): all points are negative except $x_i$.

It is impossible to distinguish these cases without seeing *all* the labels.[1] ▋

*Remark.*  To rephrase this example in terms of learning a linear separator with error $\leq \epsilon$, suppose the input distribution $P(\mathcal{X})$ is a density over the perimeter of the unit circle. No matter what this density is, there are always target hypotheses in $H$ which force us to ask for $\Omega(1/\epsilon)$ labels: no improvement over the sample complexity of supervised learning.

In this example, the bad target hypotheses have a large imbalance in probability mass between their positive and negative regions. By adding an extra dimension and an extra point, exactly the same example can be modified to make the bad hypotheses balanced.

Let's return to the original 2-d case. Some hypotheses must lie at depth $m$ in any query tree; but what about the rest? Well, suppose for convenience that $x_1, \ldots, x_m$ are in clockwise order around the unit circle. Then $\widehat{H} = \{h_{ij} : 1 \leq i \neq j \leq m\} \cup \{h_0, h_1\}$, where $h_{ij}$ labels $x_i \cdots x_{j-1}$ positive (if $j < i$ it wraps around) and the remaining points negative, and $h_0, h_1$ are everywhere negative/positive. It is possible to construct a query tree in which each $h_{ij}$ lies at depth $\leq 2(m/|j - i| + \log|j - i|)$. Thus, if the target hypothesis is chosen uniformly from $\widehat{H}$, the *expected* number of labels queried is at most

$$\frac{1}{m(m-1)+2}\left\{2m + \sum_{i \neq j} 2(m/|j - i| + \log|j - i|)\right\} = O(\log m).$$

This is why we place our hopes in an average-case analysis.

## 4 Main result

Let $\pi$ be any distribution over $\widehat{H}$; we will analyze search strategies according to the number of labels they require, averaged over target hypotheses drawn from $\pi$. In terms of query trees, this is the average depth of a leaf chosen according to $\pi$. Specifically, let $T$ be any tree whose leaves include the support of $\pi$. The quality of this tree is

$$Q(T, \pi) \;=\; \sum_{h \in \widehat{H}} \pi(h) \cdot (\# \text{ labels needed for } h) \;=\; \sum_{h \in \widehat{H}} \pi(h) \cdot \text{leaf-depth}(h).$$

Is there always a tree of average depth $o(m)$? The answer, once again, is sadly no.

**Claim 2** *Pick any $d \geq 2$ and any $m \geq 2d$. There is an input space $\mathcal{X}$ of size $m$ and a hypothesis class $H$ of VC dimension $d$, defined on domain $\mathcal{X}$, with the following property: if $\pi$ is chosen to be uniform over $H = \widehat{H}$, then any query tree $T$ has $Q(T, \pi) \geq m/8$.*

*Proof.* Let $\mathcal{X}$ consist of any $m$ points $x_1, \ldots, x_m$, and let $H$ consist of all hypotheses $h : \mathcal{X} \to \{0, 1\}$ which are positive on exactly $d$ inputs. In order to identify a particular element $h \in H$, any querying method must discover exactly the $d$ points $x_i$ on which $h$ is nonzero. By construction, the order in which queries are asked is irrelevant – it might as well be $x_1, x_2, \ldots$. The rest is a simple probability calculation. ∎

In our average-case model, we have seen one example in which intelligent querying results in an exponential improvement in the number of labels required, and one in which it is no help at all. Is there some generic scheme which *always* comes close to minimizing the number of queries, whatever the minimum number might be? Here's a natural candidate:

> **Greedy strategy.** Let $S \subseteq \widehat{H}$ be the current version space. For each unlabeled $x_i$, let $S_i^+$ be the hypotheses which label $x_i$ positive and $S_i^-$ the ones which label it negative. Pick the $x_i$ for which these sets are most nearly equal in $\pi$-mass, that is, for which $\min\{\pi(S_i^+), \pi(S_i^-)\}$ is largest.

We show this is almost as good at minimizing queries as *any* other strategy.

**Theorem 3** *Let $\pi$ be any distribution over $\widehat{H}$. Suppose that the optimal query tree requires $Q^*$ labels in expectation, for target hypotheses chosen according to $\pi$. Then the expected number of labels needed by the greedy strategy is at most $4Q^* \ln 1/(\min_h \pi(h))$.*

For the case of uniform $\pi$, the approximation ratio is thus at most $4 \ln |\widehat{H}|$. We also show almost-matching lower bounds in both the uniform and non-uniform cases.

## 5 Analysis of the greedy active learner

### 5.1 Lower bounds on the greedy scheme

The greedy approach is not optimal because it doesn't take into account the way in which a query reshapes the search space – specifically, the effect of a query on the quality of *other* queries. For instance, $\widehat{H}$ might consist of several dense clusters, each of which permits rapid binary search. However, the version space must first be whittled down to one of these subregions, and this process, though ultimately optimal, might initially be slower at shrinking the hypothesis space than more shortsighted alternatives. A concrete example of this type gives rise to the following lower bound.

**Claim 4** *For any $n \geq 16$ which is a power of two, there is a concept class $\widehat{H}_n$ of size $n$ such that: under uniform $\pi$, the optimal tree has average height at most $q_n = \Theta(\log n)$, but the greedy active learning strategy produces a tree of average height $\Omega(q_n \cdot \frac{\log n}{\log \log n})$.*

For non-uniform $\pi$, the greedy scheme can deviate more substantially from optimality.

**Claim 5** *For any $n \geq 2$, there is a hypothesis class $\widehat{H}$ with $2n + 1$ elements and a distribution $\pi$ over $\widehat{H}$, such that: (a) $\pi$ ranges in value from $1/2$ to $1/2^{n+1}$; (b) the optimal tree has average depth less than 3; (c) the greedy tree has average depth at least $n/2$.*

Proofs of these lower bounds appear in the full paper, available at the author's website.

## 5.2 Upper bound

**Overview.** The lower bounds on the quality of a greedy learner are sobering, but things cannot get too much worse than this. Here's the basic argument for uniform $\pi$: we show that if the optimal tree $T^*$ requires $Q^*$ queries in expectation, then some query must (again in expectation) "cut off" a chunk of $\widehat{H}$ of $\pi$-mass $\Omega(1/Q^*)$. Therefore, the root query of the greedy tree $T_G$ is at least this good (cf. Johnson's set cover analysis [8]). Things get trickier when we try to show that the rest of $T_G$ is also good, because although $T^*$ uses just $Q^*$ queries *on average*, it may need many more queries for certain hypotheses. Subtrees of $T_G$ could correspond to version spaces for which more than $Q^*$ queries are needed, and the roots of these subtrees might not cut down the version space much...

For a worst-case model, a proof of approximate optimality is known in a related context [6]; as we saw in Claim 1, that model is trivial in our situation. The average-case model, and especially the use of arbitrary weights $\pi$, require more care.

**Details.** For want of space, we only discuss some issues that arise in proving the main theorem, and leave the actual proof to the full paper.

The key concept we have to define is the *quality* of a query, and it turns out that we need this to be monotonically decreasing, that is, it should only go down as active learning proceeds and the version space shrinks. This rules out some natural entropy-based notions.

Suppose we are down to some version space $S \subseteq \widehat{H}$, and a possible next query is $x_j$. If $S^+$ is the subset of $S$ which labels $x_j$ positive, and $S^-$ are the ones that label it negative, then on average the probability mass (measured by $\pi$) eliminated by $x_j$ is

$$\frac{\pi(S^+)}{\pi(S)}\pi(S^-) + \frac{\pi(S^-)}{\pi(S)}\pi(S^+) \;\; = \;\; \frac{2\pi(S^+)\pi(S^-)}{\pi(S)}.$$

We say $x_j$ *shrinks* $(S, \pi)$ by this much, with the understanding that this is in expectation. Shrinkage is easily seen to have the monotonicity property we need.

**Lemma 6** *If $x_j$ shrinks $(\widehat{H}, \pi)$ by $\Delta$, then it shrinks $(S, \pi)$ by at most $\Delta$ for any $S \subseteq \widehat{H}$.*

We would expect that if the optimal tree is short, there must be at least one query which shrinks $(\widehat{H}, \pi)$ considerably. More concretely, the definition of shrinkage seems to suggest that if all queries provide shrinkage at most $\Delta$, and the current version space has mass $\pi(S)$, then at least about $\pi(S)/\Delta$ more queries are needed. This isn't entirely true, because of a second effect: if $|S| = 2$, then we need just one query, regardless of $\pi(S)$.

Roughly speaking, when there are lots of hypotheses with significant mass left in $S$, the first effect dominates; thereafter the second takes over. To smoothly incorporate both effects, we use the notion of *collision probability*. For a distribution $\nu$ over support $\mathcal{Z}$, this is $\mathsf{CP}(\nu) = \sum_{z \in \mathcal{Z}} \nu(z)^2$, the chance that two random draws from $\nu$ are identical.

**Lemma 7** *Suppose every query shrinks $(\widehat{H}, \pi)$ by at most $\Delta > 0$. Pick any $S \subseteq \widehat{H}$, and any query tree $T$ whose leaves include $S$. If $\pi_S$ is the restriction of $\pi$ to $S$ (that is, $\pi_S(h) = \pi(h)/\pi(S)$ for $h \in S$), then $Q(T, \pi_S) \geq (1 - \mathsf{CP}(\pi_S)) \cdot \pi(S)/\Delta$.*

**Corollary 8** *Pick any* $S \subseteq \widehat{H}$ *and any tree* $T$ *whose leaves include all of* $S$. *Then there must exist a query which shrinks* $(S, \pi_S)$ *by at least* $(1 - \mathsf{CP}(\pi_S))/Q(T, \pi_S)$.

So if the current version space $S \subseteq \widehat{H}$ is such that $\pi_S$ has small collision probability, some query must split off a sizeable chunk of $S$. This can form the basis of a proof by induction.

But what if $\mathsf{CP}(\pi_S)$ is large, say greater than $1/2$? In this case, the mass of some particular hypothesis $h_0 \in S$ exceeds that of all the others combined, and $S$ could shrink by just an insignificant amount during the subsequent greedy query, or even during the next few iterations of greedy queries. It turns out, however, that within roughly the number of iterations that the optimal tree needs for target $h_0$, the greedy procedure will either reject $h_0$ or identify it as the target. If it is rejected, then *by that time* $S$ will have shrunk considerably.

By combining the two cases for $\mathsf{CP}(\pi_S)$, we get the following lemma, which is proved in the full paper and yields our main theorem as an immediate consequence.

**Lemma 9** *Let* $T^*$ *denote any particular query tree for* $\pi$, *and let* $T$ *be the greedily-constructed query tree. For any* $S \subseteq \widehat{H}$ *which corresponds to a subtree* $T_S$ *of* $T$,

$$Q(T_S, \pi_S) \leq 4Q(T^*, \pi_S) \ln \frac{\pi(S)}{\min_{h \in S} \pi(h)}.$$

## 6   Related work and promising directions

Rather than attempting to summarize the wide range of proposed active learning methods, for instance [5, 7, 10, 13, 14], we will discuss three basic techniques upon which they rely.

**Greedy search.** This is the technique we have abstracted and rigorously validated in this paper. It is the foundation of most of the schemes cited above. Algorithmically, the main problem is that the query selection rule is not immediately tractable, so approximations are necessary. For linear separators, $\widehat{H}$ consists of convex sets, and if $\pi$ is chosen to be proportional to volume, query selection involves estimating volumes of convex regions, which is tractable but (using present techniques) inconvenient. Tong and Koller [13] investigate margin-based approximations which are efficiently computable using SVM technology.

**Opportunistic priors.** This is a trick in which the learner takes a look at the unlabeled data and then places bets on hypotheses. A uniform bet over all of $\widehat{H}$ leads to standard generalization bounds. But if the algorithm places more weight on certain hypotheses (for instance, those with large margin), then its final error bound is excellent if it guessed right, and worse-than-usual if it guessed wrong. This technique is not specific to active learning, and has been analyzed elsewhere (eg. [12]). One interesting line of work investigates a flexible family of priors specified by pairwise similarities between data points, eg. [14].

**Bayesian assumptions.** In our analysis, although $\pi$ can be seen as some sort of prior belief, there is no assumption that nature shares this belief; in particular, the generalization bound does not depend on it. A Bayesian assumption has an immediate benefit for active learning: if at any stage the remaining version space (weighted by prior $\pi$) is largely in agreement on the unlabeled data, it is legitimate to stop and output one of these remaining hypotheses [7]. In a non-Bayesian setting this is not legitimate.

When the hypothesis class consists of probabilistic classifiers, the Bayesian assumption has also been used in another way: to approximate the greedy selection rule using the MAP estimate instead of an expensive summation over the posterior (eg. [11]).

In terms of theoretical results, another work which considers the tradeoff between labels and generalization error is [7], in which a greedy scheme, realized using sampling, is analyzed in a Bayesian setting. The authors show that it is possible to achieve an exponential

improvement in the number of labels needed to learn linear separators, when both data and target hypothesis are chosen uniformly from the unit sphere. It is an intriguing question whether this holds for more general data distributions.

**Other directions.** We have looked at the case where the acceptable error rate is fixed and the goal is to minimize the number of queries. What about fixing the number of queries and asking for the best (average) error rate possible? In other words, the query tree has a fixed depth, and each leaf is annotated with its remaining version space $S \subseteq \widehat{H}$. Treating each element of $S$ as a point in $\{0, 1\}^m$ (its predictions on the pool of data), the error at this leaf depends on the Hamming diameter of $S$. What is a good querying strategy for producing low-diameter leaves?

The most widely-used classifiers are perhaps linear separators. Existing active learning schemes ignore the rich algebraic structure of $\widehat{H}$, an *arrangement of hyperplanes* [4].

**Acknowledgements.** I am very grateful to the anonymous NIPS reviewers for their careful and detailed feedback.

## Footnotes

[1]What if the final hypothesis – considered as a point in $\{0,1\}^m$ – doesn't have to be exactly right, but within Hamming distance $k$ of the correct one? Then a similar example forces $\Omega(m/k)$ queries.

# References

[1] D. Angluin. Queries and concept learning. *Machine Learning*, 2:319–342, 1988.

[2] D. Angluin. Queries revisited. *Proceedings of the Twelfth International Conference on Algorithmic Learning Theory*, pages 12–31, 2001.

[3] E.B. Baum and K. Lang. Query learning can work poorly when a human oracle is used. *International Joint Conference on Neural Networks*, 1992.

[4] A. Bjorner, M. Las Vergnas, B. Sturmfels, N. White, and G. Ziegler. *Oriented matroids*. Cambridge University Press, 1999.

[5] D. Cohn, Z. Ghahramani, and M. Jordan. Active learning with statistical models. *Journal of Artificial Intelligence Research*, 4:129–145, 1996.

[6] S. Dasgupta, P.M. Long, and W.S. Lee. A theoretical analysis of query selection for collaborative fi ltering. *Machine Learning*, 51:283–298, 2003.

[7] Y. Freund, S. Seung, E. Shamir, and N. Tishby. Selective sampling using the query by committee algorithm. *Machine Learning*, 28:133–168, 1997.

[8] D.S. Johnson. Approximation algorithms for combinatorial problems. *Journal of Computer and System Sciences*, 9:256–278, 1974.

[9] M.J. Kearns and U.V. Vazirani. *An introduction to computational learning theory*. MIT Press, 1993.

[10] A. McCallum and K. Nigam. Employing em and pool-based active learning for text classification. *Fifteenth International Conference on Machine Learning*, 1998.

[11] N. Roy and A. McCallum. Toward optimal active learning through sampling of error reduction. *Twentieth International Conference on Machine Learning*, 2003.

[12] J. Shawe-Taylor, P. Bartlett, R. Williamson, and M. Anthony. Structural risk minimization over data-dependent hierarchies. *IEEE Transactions on Information Theory*, 1998.

[13] S. Tong and D. Koller. Support vector machine active learning with applications to text classification. *Journal of Machine Learning Research*, 2001.

[14] X. Zhu, J. Lafferty, and Z. Ghahramani. Combining active learning and semi-supervised learning using gaussian fi elds and harmonic functions. *ICML workshop*, 2003.
